# Exploratory Data Analysis Using Radial Basis Function Latent Variable Models

**Alan D. Marrs and Andrew R. Webb**
DERA
St Andrews Road, Malvern
Worcestershire U.K. WR14 3PS
{marrs,webb}@signal.dera.gov.uk

## Abstract

Two developments of nonlinear latent variable models based on radial basis functions are discussed: in the first, the use of priors or constraints on allowable models is considered as a means of preserving data structure in low-dimensional representations for visualisation purposes. Also, a resampling approach is introduced which makes more effective use of the latent samples in evaluating the likelihood.

## 1 INTRODUCTION

Radial basis functions (RBF) have been extensively used for problems in discrimination and regression. Here we consider their application for obtaining low–dimensional representations of high–dimensional data as part of the exploratory data analysis process. There has been a great deal of research over the years into linear and nonlinear techniques for dimensionality reduction. The technique most commonly used is principal components analysis (PCA) and there have been several nonlinear generalisations, each taking a particular definition of PCA and generalising it to the nonlinear situation.

One approach is to find surfaces of closest fit (as a generalisation of the PCA definition due to the work of Pearson (1901) for finding lines and planes of closest fit). This has been explored by Hastie and Stuetzle (1989), Tibshirani (1992) (and further by LeBlanc and Tibshirani, 1994) and various authors using a neural network approach (for example, Kramer, 1991). Another approach is one of variance maximisation subject to constraints on the transformation (Hotelling, 1933). This has been investigated by Webb (1996), using a transformation modelled as an RBF network, and in a supervised context in Webb (1998).

An alternative strategy also using RBFs, based on metric multidimensional scaling, is described by Webb (1995) and Lowe and Tipping (1996). Here, an optimisation criterion,

termed *stress*, is defined in the transformed space and the weights in an RBF model determined by minimising the stress.

The above methods use a radial basis function to model a transformation from the high–dimensional *data space* to a low–dimensional *representation space*. A complementary approach is provided by Bishop *et al* (1998) in which the structure of the data is modelled as a function of hidden or latent variables. Termed *generative topographic mapping* (GTM), the model may be regarded as a nonlinear generalisation of factor analysis in which the mapping from latent space to data space is characterised by an RBF.

Such generative models are relevant to a wide range of applications including radar target modelling, speech recognition and handwritten character recognition.

However, one of the problems with GTM that limits its practical use for visualising data on manifolds in high dimensional space arises from distortions in the structure that it imposes. This is acknowledged in Bishop *et al* (1997) where 'magnification factors' are introduced to correct for the GTM's deficiency as a means of data visualisation.

This paper considers two developments: constraints on the permissible models and resampling of the latent space. Section 2 presents the background to latent variable models; Model constraints are discussed in Section 3. Section 4 describes a re–sampling approach to estimation of the posterior pdf on the latent samples. An illustration is provided in Section 5.

## 2   BACKGROUND

Briefly, we shall re–state the basic GTM model, retaining the notation of Bishop *et al* (1998). Let $\{t_i, i = 1, \ldots, N\}, t_i \in \mathbb{R}^D$ represent measurements on the data space variables; $x \in \mathbb{R}^L$ represent the latent variables.

Let $t$ be normally–distributed with mean $y(x; W)$ and covariance matrix $\beta^{-1}I$; $y(x; W)$ is a nonlinear transformation that depends on a set of parameters $W$. Specifically, we shall assume a basis function model

$$y(x; W) = \sum_{i=1}^{M} w_i \phi_i(x)$$

where the vectors $w_i \in \mathbb{R}^D$ are to be determined through optimisation and $\{\phi_i, i = 1, \ldots, M\}$ is a set of basis functions defined on the latent space.

The data distribution may be written

$$p(t|W, \beta) = \int p(t|x; W, \beta) p(x) dx \qquad (1)$$

where, under the assumptions of normality,

$$p(t|x; W, \beta) = \left(\frac{\beta}{2\pi}\right)^{D/2} \exp\left\{-\frac{\beta}{2}\|y(x; W) - t\|^2\right\}$$

Approximating the integral by a finite sum (assuming the functions $p(x)$ and $y(x)$ do not vary too greatly compared with the sample spacing), we have

$$p(t|W, \beta) = \sum_{i=1}^{K} p_i p(t|x_i; W, \beta) \qquad (2)$$

which may be regarded as a function of the parameters $W$ and $\beta$ that characterise $y$.

Given the data set $\{t_i, i = 1, \ldots, N\}$, the log likelihood is given by

$$L(\boldsymbol{W}, \beta) = \sum_{j=1}^{N} \ln[p(t_j | \boldsymbol{W}, \beta)]$$

which may be maximised using a standard EM approach (Bishop *et al*, 1998).

In this case, we have

$$\hat{p}_j = \frac{1}{N} \sum_{n=1}^{N} R_{in} \qquad (3)$$

as the re–estimate of the mixture component weights, $p_j$, at the $(m+1)$ step, where

$$R_{in} = \frac{p_i^{(m)} p(t_n | \boldsymbol{x}_i; \boldsymbol{W}^{(m)}, \beta^{(m)})}{\sum_i p_i^{(m)} p(t_n | \boldsymbol{x}_i; \boldsymbol{W}^{(m)}, \beta^{(m)})} \qquad (4)$$

and $(.)^{(m)}$ denotes values at the $m$th step. Note that Bishop *et al* (1998) do not re–estimate $p_j$; all values are taken to be equal.

The number of $p_j$ terms to be re–estimated is $K$, the number of terms used to approximate the integral (1). We might expect that the density is smoothly varying and governed by a much fewer number of parameters (not dependent on $K$).

The re–estimation equation for the $D \times M$ matrix $\boldsymbol{W} = [\boldsymbol{w}_1 | \ldots | \boldsymbol{w}_M]$ is

$$\boldsymbol{W}^{(m+1)} = \boldsymbol{T}^T \boldsymbol{R}^T \boldsymbol{\Phi} [\boldsymbol{\Phi}^T \boldsymbol{G} \boldsymbol{\Phi}]^{-1} \qquad (5)$$

where $\boldsymbol{G}$ is the $K \times K$ diagonal matrix with

$$G_{jj} = \sum_{n=1}^{N} R_{jn}$$

and $\boldsymbol{T}^T = [\boldsymbol{t}_1 | \ldots | \boldsymbol{t}_N]$, $\boldsymbol{\Phi}^T = [\phi(\boldsymbol{x}_1) | \ldots | \phi(\boldsymbol{x}_K)]$. The term $\beta$ is re–estimated as $1/\beta^{(m)} = 1/(ND) \sum_{i=1}^{N} \sum_{j=1}^{K} R_{ji} |t_i - \boldsymbol{W}^{(m+1)} \phi(\boldsymbol{x}_j)|^2$.

Once we have determined parameters of the transformation, we may invert the model by asking for the distribution of $\boldsymbol{x}$ given a measurement $t_i$. That is, we require

$$p(\boldsymbol{x} | t_i) = \frac{p(t_i | \boldsymbol{x}) p(\boldsymbol{x})}{\int p(t_i | \boldsymbol{x}) p(\boldsymbol{x}) d\boldsymbol{x}} \qquad (6)$$

For example, we may plot the position of the peak of the distribution $p(\boldsymbol{x} | t_i)$ for each data sample $t_i$.

## 3  APPLYING A CONSTRAINT

One way to retain structure is to impose a condition that ensures that a unit step in the latent space corresponds to a unit step in the data space (more or less). For a single latent variable, $x_1$, we may impose the constraints that

$$\left| \frac{\partial y}{\partial x_1} \right|^2 = 1$$

which may be written, in terms of $\boldsymbol{W}$ as

$$j_1^T \boldsymbol{W}^T \boldsymbol{W} j_1 = 1$$

where $j_1 = \frac{\partial \phi}{\partial x_1}$.

The derivative of the data space variable with respect to the latent variable has unit magnitude. The derivative is of course a function of $x_1$ and imposing such a condition at each sample point in latent space would not be possible owing to the smoothness of the RBF model. However, we may average over the latent space,

$$\overline{\left|\frac{\partial y}{\partial x_1}\right|^2} = 1$$

where $\overline{(.)}$ denotes average over the latent space.

In general, for $L$ latent variables we may impose a constraint $\overline{J^T W^T W J} = I_L$ leading to the penalty term

$$\mathrm{Tr}\left\{\Lambda(\overline{J^T W^T W J} - I_L)\right\}$$

where $J$ is an $M \times L$ matrix with $j$th column $\partial \phi / \partial x_j$ and $\Lambda$ is a symmetric matrix of Lagrange multipliers. This is very similar to regularisation terms. It is a condition on the norm of $W$; it incorporates the Jacobian matrix $J$ and a symmetric $L \times L$ matrix of Lagrange multipliers, $\Lambda$. The re–estimation solution for $W$ may be written

$$W = T^T R^T \Phi (\Phi^T G \Phi + \overline{J\Lambda J^T})^{-1} \tag{7}$$

with $\Lambda$ chosen so that the constraint $\overline{J^T W^T W J} = I_L$ is satisfied.

We may also use the derivatives of the transformation to define a distortion measure or *magnification factor*,

$$M(x; W) = \|J^T W^T W J - I\|^2$$

which is a function of the latent variables and the model parameters. A value of zero shows that there is no distortion[1].

An alternative to the constraint approach above is to introduce a prior on the allowable transformations using the magnification factor; for example,

$$P(W) \approx \exp(-\lambda \overline{M(x; W)}) \tag{8}$$

where $\lambda$ is a regularisation parameter. This leads to a modification to the M–step re-estimation equation for $W$, providing a maximum a posteriori estimate. Equation (8) provides a natural generalisation of PCA since for the special case of a linear transformation ($\phi_i = x_i$, $M = L$), the solution for $W$ is the PCA space as $\lambda \rightarrow \infty$.

## 4   RESAMPLING THE LATENT SPACE

Having obtained a mapping from latent space to data space using the above constraint, we seek a better estimate to the posterior pdf of the latent samples. Current versions of GTM require the latent samples to be uniformly distributed in the latent space which leads to distortions when the data of interest are projected into the latent space for visualisation. Since the responsibility matrix $R$ can be used to determine a weight for each of the latent samples it is possible to update these samples using a resampling scheme.

We propose to use a resampling scheme based upon adaptive kernel density estimation. The basic procedure places a Gaussian kernel on each latent sample. This results in a Gaussian

mixture representation of the pdf of the latent samples $p(x|t)$,

$$p(x|t) = \sum_{i=1}^{K} p_i N(\mu_i, \Sigma_i),$$ (9)

where each mixture component is weighted according to the latent sample weight $p_i$. Initially, the $\Sigma_i$'s are all equal, taking their value from the standard formula of Silverman (1986),

$$\Sigma_i = h^L V,$$ (10)

where matrix $V$ is an estimate of the covariance of $p(x)$ and,

$$h = [\{4/(L+2)\}^{1/(L+4)} K^{-1/(L+4)}]^L.$$ (11)

If the kernels are centered exactly on the latent samples, this model artificially inflates the variance of the latent samples. Following West (1993) we perform kernel shrinkage by making the $\mu_i$ take the values

$$\mu_i = \sqrt{1 - h^L} x_i + (1 - \sqrt{1 - h^L})\hat{\mu}$$ (12)

where $\hat{\mu}$ is the mean of the latent samples. This ensures that there is no artificial inflation of the variance.

To reduce the redundancy in our initially large number of mixture components, we propose a kernel reduction scheme in a similar manner to West. However, the scheme used here differs from that of West and follows a scheme proposed by Salmond (1990). Essentially, we chose the component with the smallest weight and its nearest neighbour, denoting these with subscripts 1 and 2 respectively. These components are then combined into a single component denoted with subscript $c$ as follows,

$$p_c = p_1 + p_2$$ (13)

$$\mu_c = \frac{p_1 \mu_1 + p_2 \mu_2}{p_c}$$ (14)

$$\Sigma_c = \frac{p_1[\Sigma_1 + (\mu_c - \mu_1)(\mu_c - \mu_1)^T] + p_2[\Sigma_2 + (\mu_c - \mu_2)(\mu_c - \mu_2)^T]}{p_c}.$$ (15)

This procedure is repeated until some stopping criterion is met. The stopping criterion could be a simple limit upon the number of mixture components ie; smaller than $K$ but sufficiently large to model the data structure. Alternatively, the average kernel covariance and between kernel covariance can be monitored and the reduction stopped before some multiple (eg. 10) of the average kernel covariance exceeds the between kernel covariance.

Once a final mixture density estimate is obtained, a new set of equally weighted latent samples can be drawn from it. The new latent samples represent a better estimate of the posterior pdf of the latent samples and can be used, along with the existing RBF mapping, to calculate a new responsibility matrix $R$. This procedure can be repeated to obtain a further improved estimate of the posterior pdf which, after only a couple of iterations can lead to good estimates of the posterior pdf which further iterations fail to improve upon.

## 5  RESULTS

A latent variable model based on a spherically–symmetric Gaussian RBF has been implemented. The weights and the centres of the RBF were initialised so that the solution best approximated the zero–distortion principal components solution for two–dimensional projection.

For our example we chose to construct a simulated data set with easily identifiable structure. Four hundred points lying on the letters "NIPS" were sampled and projected onto a sphere of radius 50 such that the points lay between $25°$ and $175°$ longitude and $75°$ and $125°$ latitude with Gaussian noise of variance 4.0 on the radius of each point. The resulting data are shown in figure 1.

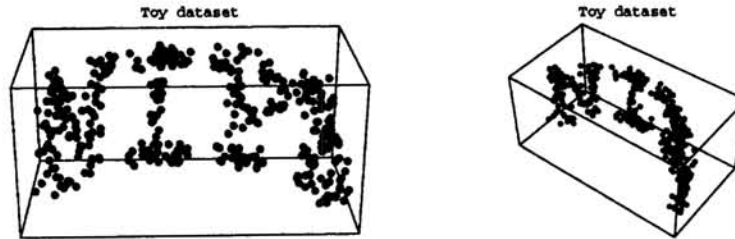

Figure 1: Simulated data.

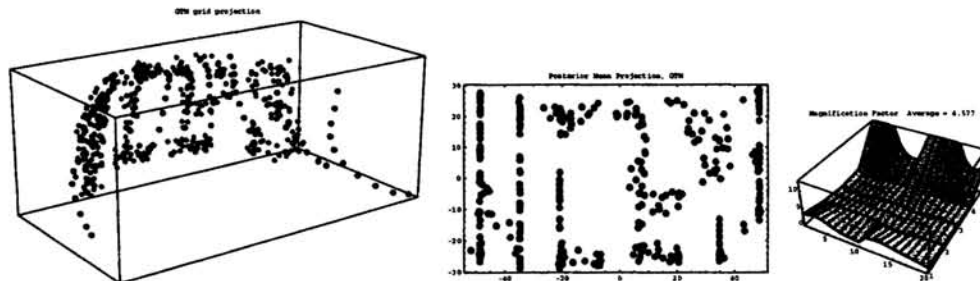

Figure 2: Results for standard GTM model.

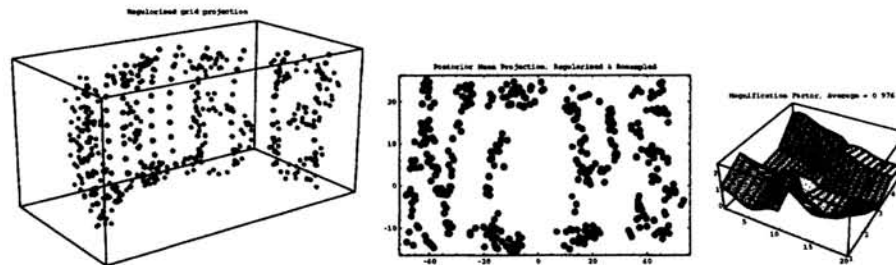

Figure 3: Results for regularised/resampled model.

Figure 2 shows results for the standard GTM (uniform grid of latent samples) projection of the data to two dimensions. The central figure shows the projection onto the latent space, exhibiting significant distortion. The left figure shows the projection of the regular grid of latent samples (red points) into the data space. Distortion of this grid can be easily seen. The right figure is a plot of the magnification factor as defined in section 3, with mean value of 4.577. For this data set most stretching occurs at the edges of the latent variable space.

Figure 3 shows results for the regularised/resampled version of the latent variable model for $\lambda = 1.0$. Again the central figure shows the projection onto the latent space after 2 iterations of the resampling procedure. The left-hand figure shows the projection of the initial regular grid of latent samples into the data space. The effect of regularisation is evident by the lack of severe distortions. Finally the magnification factors can be seen in the right-hand figure to be lower, with a mean value of 0.976.

# 6 DISCUSSION

We have considered two developments of the GTM latent variable model: the incorporation of priors on the allowable model and a resampling approach to the maximum likelihood parameter estimation. Results have been presented for this regularised/resampling approach and magnification factors lower than the standard model achieved, using the same RBF model. However, further reduction in magnification factor is possible with different RBF models, but the example illustrates that resampling offers a more robust approach. Current work is aimed at assessing the approach on realistic data sets.

## Footnotes

[1]Note that this differs from the measure in the paper by Bishop *et al*, where a ratio–of–areas criterion is used, a factor which is unity for zero distortion, but may also be unity for some distortions.

## References

Bishop, C.M. and Svensén, M. and Williams, C.K.I. (1997). Magnification factors for the GTM algorithm. *IEE International Conference on Artificial Neural Networks*, 465–471.

Bishop, C.M. and Svensén, M. and Williams, C.K.I. (1998). GTM: the generative topographic mapping. *Neural Computation*, 10, 215–234.

Hastie, T. and Stuetzle, W. (1989). Principal curves, *Journal of the American Statistical Association*, 84, 502–516.

Hotelling, H. (1933). Analysis of a complex of statistical variables into principal components. *Journal of Educational Psychology*, 24, 417–441, 498–520.

Kramer, M.A. (1991). Nonlinear principal component analysis using autoassociative neural networks. *American Institute of Chemical Engineers Journal*, 37(2), 233–243.

LeBlanc, M. and Tibshirani, R. (1994). Adaptive principal surfaces. *Journal of the American Statistical Association*, 89(425), 53–664.

Lowe, D. and Tipping, M. (1996). Feed–forward neural networks and topographic mappings for exploratory data analysis. *Neural Computing and Applications*, 4, 83–95.

Pearson, K. (1901). On lines and planes of closest fit. *Philosophical Magazine*, 6, 559–572.

Salmond, D.J. (1990). Mixture reduction algorithms for target tracking in clutter. *Signal & Data processing of small targets, edited by O. Drummond, SPIE*, 1305.

Silverman, B.W. (1986). Density Estimation for Statistics and Data Analysis. *Chapman & Hall*, 1986.

Tibshirani, R. (1992). Principal curves revisited. *Statistics and Computing*, 2(4), 183–190.

Webb, A.R. (1995). Multidimensional scaling by iterative majorisation using radial basis functions. Pattern Recognition, 28(5), 753-759.

Webb, A.R. (1996). An approach to nonlinear principal components analysis using radially-symmetric kernel functions. *Statistics and Computing*, 6, 159-168.

Webb, A.R. (1997). Radial basis functions for exploratory data analysis: an iterative majorisation approach for Minkowski distances based on multidimensional scaling. *Journal of Classification*, 14(2), 249-267.

Webb, A.R. (1998). Supervised nonlinear principal components analysis. (submitted for publication).

West, M. (1993). Approximating posterior distributions by mixtures. *J. R. Statist. Soc B*, 55(2), 409-422.
